# Learning in Silicon: Timing is Everything

**John V. Arthur and Kwabena Boahen**
Department of Bioengineering
University of Pennsylvania
Philadelphia, PA 19104
{jarthur, boahen}@seas.upenn.edu

## Abstract

We describe a neuromorphic chip that uses binary synapses with spike timing-dependent plasticity (STDP) to learn stimulated patterns of activity and to compensate for variability in excitability. Specifically, STDP preferentially potentiates (turns on) synapses that project from excitable neurons, which spike early, to lethargic neurons, which spike late. The additional excitatory synaptic current makes lethargic neurons spike earlier, thereby causing neurons that belong to the same pattern to spike in synchrony. Once learned, an entire pattern can be recalled by stimulating a subset.

## 1 Variability in Neural Systems

Evidence suggests precise spike timing is important in neural coding, specifically, in the hippocampus. The hippocampus uses timing in the spike activity of place cells (in addition to rate) to encode location in space [1]. Place cells employ a *phase code*: the timing at which a neuron spikes relative to the phase of the inhibitory theta rhythm (5-12Hz) conveys information. As an animal approaches a place cell's preferred location, the place cell not only increases its spike rate, but also spikes at earlier phases in the theta cycle.

To implement a phase code, the theta rhythm is thought to prevent spiking until the input synaptic current exceeds the sum of the neuron threshold and the decreasing inhibition on the downward phase of the cycle [2]. However, even with identical inputs and common theta inhibition, neurons do not spike in synchrony. Variability in excitability spreads the activity in phase. Lethargic neurons (such as those with high thresholds) spike late in the theta cycle, since their input exceeds the sum of the neuron threshold and theta inhibition only after the theta inhibition has had time to decrease. Conversely, excitable neurons (such as those with low thresholds) spike early in the theta cycle. Consequently, variability in excitability translates into variability in timing.

We hypothesize that the hippocampus achieves its precise spike timing (about 10ms) through *plasticity enhanced phase-coding* (PEP). The source of hippocampal timing precision in the presence of variability (and noise) remains unexplained. Synaptic plasticity can compensate for variability in excitability if it increases excitatory synaptic input to neurons in inverse proportion to their excitabilities. Recasting this in a phase-coding framework, we desire a learning rule that increases excitatory synaptic input to neurons directly related to their phases. Neurons that lag require additional synaptic input, whereas neurons that lead

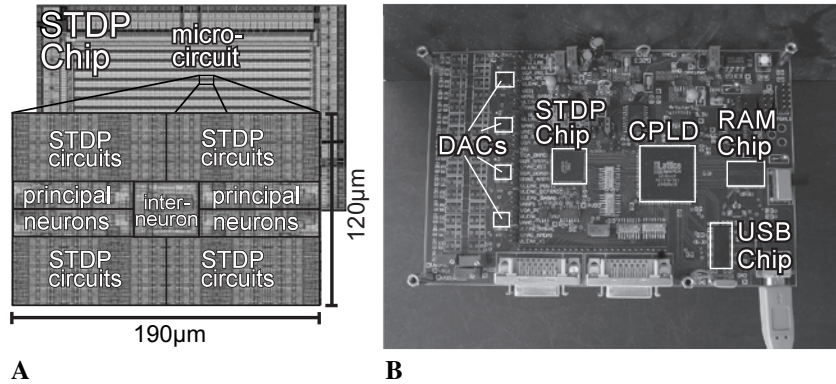

**A**                                    **B**

Figure 1: STDP Chip. **A** The chip has a 16-by-16 array of microcircuits; one microcircuit includes four principal neurons, each with 21 STDP circuits. **B** The STDP Chip is embedded in a circuit board including DACs, a CPLD, a RAM chip, and a USB chip, which communicates with a PC.

require none. The spike timing-dependent plasticity (STDP) observed in the hippocampus satisfies this requirement [3]. It requires repeated pre-before-post spike pairings (within a time window) to potentiate and repeated post-before-pre pairings to depress a synapse.

Here we validate our hypothesis with a model implemented in silicon, where variability is as ubiquitous as it is in biology [4]. Section 2 presents our silicon system, including the STDP Chip. Section 3 describes and characterizes the STDP circuit. Section 4 demonstrates that PEP compensates for variability and provides evidence that STDP is the compensation mechanism. Section 5 explores a desirable consequence of PEP: unconventional associative pattern recall. Section 6 discusses the implications of the PEP model, including its benefits and applications in the engineering of neuromorphic systems and in the study of neurobiology.

## 2   Silicon System

We have designed, submitted, and tested a silicon implementation of PEP. The STDP Chip was fabricated through MOSIS in a 1P5M $0.25\mu$m CMOS process, with just under 750,000 transistors in just over $10\text{mm}^2$ of area. It has a 32 by 32 array of excitatory principal neurons commingled with a 16 by 16 array of inhibitory interneurons that are not used here (Figure 1A). Each principal neuron has 21 STDP synapses. The address-event representation (AER) [5] is used to transmit spikes off chip and to receive afferent and recurrent spike input.

To configure the STDP Chip as a recurrent network, we embedded it in a circuit board (Figure 1B). The board has five primary components: a CPLD (complex programmable logic device), the STDP Chip, a RAM chip, a USB interface chip, and DACs (digital-to-analog converters). The central component in the system is the CPLD. The CPLD handles AER traffic, mediates communication between devices, and implements recurrent connections by accessing a lookup table, stored in the RAM chip. The USB interface chip provides a bidirectional link with a PC. The DACs control the analog biases in the system, including the leak current, which the PC varies in real-time to create the global inhibitory theta rhythm.

The principal neuron consists of a refractory period and calcium-dependent potassium circuit (RCK), a synapse circuit, and a soma circuit (Figure 2A). RCK and the synapse are

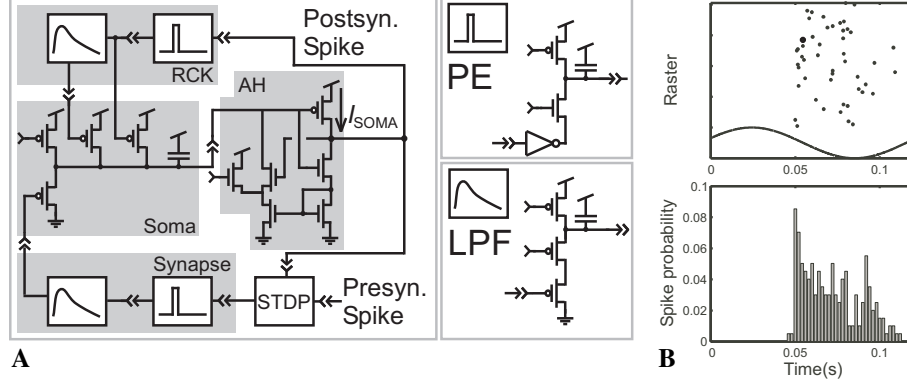

Figure 2: Principal neuron. **A** A simplified schematic is shown, including: the synapse, refractory and calcium-dependent potassium channel (RCK), soma, and axon-hillock (AH) circuits, plus their constituent elements, the pulse extender (PE) and the low-pass filter (LPF). **B** Spikes (dots) from 81 principal neurons are temporally dispersed, when excited by poisson-like inputs (58Hz) and inhibited by the common 8.3Hz theta rhythm (solid line). The histogram includes spikes from five theta cycles.

composed of two reusable blocks: the low-pass filter (LPF) and the pulse extender (PE). The soma is a modified version of the LPF, which receives additional input from an axon-hillock circuit (AH).

RCK is inhibitory to the neuron. It consists of a PE, which models calcium influx during a spike, and a LPF, which models calcium buffering. When AH fires a spike, a packet of charge is dumped onto a capacitor in the PE. The PE's output activates until the charge decays away, which takes a few milliseconds. Also, while the PE is active, charge accumulates on the LPF's capacitor, lowering the LPF's output voltage. Once the PE deactivates, this charge leaks away as well, but this takes tens of milliseconds because the leak is smaller. The PE's and the LPF's inhibitory effects on the soma are both described below in terms of the sum ($I_{SHUNT}$) of the currents their output voltages produce in pMOS transistors whose sources are at Vdd (see Figure 2A). Note that, in the absence of spikes, these currents decay exponentially, with a time-constant determined by their respective leaks.

The synapse circuit is excitatory to the neuron. It is composed of a PE, which represents the neurotransmitter released into the synaptic cleft, and a LPF, which represents the bound neurotransmitter. The synapse circuit is similar to RCK in structure but differs in function: It is activated not by the principal neuron itself but by the STDP circuits (or directly by afferent spikes that bypass these circuits, i.e., fixed synapses). The synapse's effect on the soma is also described below in terms of the current ($I_{SYN}$) its output voltage produces in a pMOS transistor whose source is at Vdd.

The soma circuit is a leaky integrator. It receives excitation from the synapse circuit and shunting inhibition from RCK and has a leak current as well. Its temporal behavior is described by:

$$\tau \frac{dI_{SOMA}}{dt} + I_{SOMA} = \frac{I_{SYN} I_0}{I_{SHUNT}}$$

where $I_{SOMA}$ is the current the capacitor's voltage produces in a pMOS transistor whose source is at Vdd (see Figure 2A). $I_{SHUNT}$ is the sum of the leak, refractory, and calcium-dependent potassium currents. These currents also determine the time constant: $\tau = \frac{C U_t}{\kappa I_{SHUNT}}$, where $I_0$ and $\kappa$ are transistor parameters and $U_t$ is the thermal voltage.

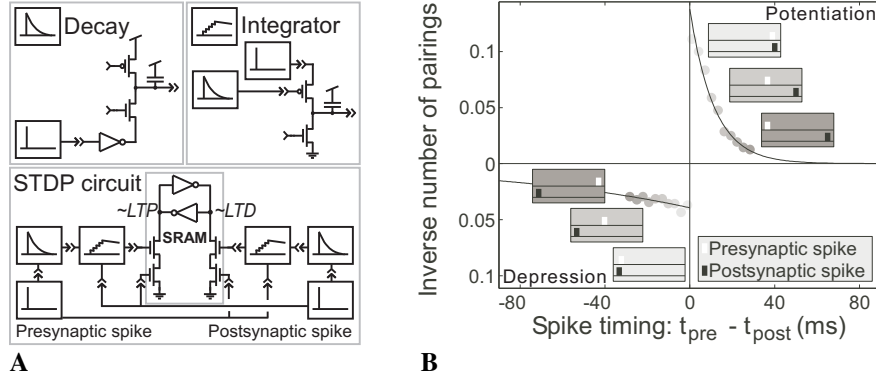

Figure 3: STDP circuit design and characterization. **A** The circuit is composed of three subcircuits: decay, integrator, and SRAM. **B** The circuit potentiates when the presynaptic spike precedes the postsynaptic spike and depresses when the postsynaptic spike precedes the presynaptic spike.

The soma circuit is connected to an AH, the locus of spike generation. The AH consists of model voltage-dependent sodium and potassium channel populations (modified from [6] by Kai Hynna). It initiates the AER signaling process required to send a spike off chip.

To characterize principal neuron variability, we excited 81 neurons with poisson-like 58Hz spike trains (Figure 2B). We made these spike trains poisson-like by starting with a regular 200Hz spike train and dropping spikes randomly, with probability of 0.71. Thus spikes were delivered to neurons that won the coin toss in synchrony every 5ms. However, neurons did not lock onto the input synchrony due to filtering by the synaptic time constant (see Figure 2B). They also received a common inhibitory input at the theta frequency (8.3Hz), via their leak current. Each neuron was prevented from firing more than one spike in a theta cycle by its model calcium-dependent potassium channel population.

The principal neurons' spike times were variable. To quantify the spike variability, we used timing precision, which we define as twice the standard deviation of spike times accumulated from five theta cycles. With an input rate of 58Hz the timing precision was 34ms.

## 3   STDP Circuit

The STDP circuit (related to [7]-[8]), for which the STDP Chip is named, is the most abundant, with 21,504 copies on the chip. This circuit is built from three subcircuits: decay, integrator, and SRAM (Figure 3A). The decay and integrator are used to implement potentiation, and depression, in a symmetric fashion. The SRAM holds the current binary state of the synapse, either potentiated or depressed.

For potentiation, the decay remembers the last presynaptic spike. Its capacitor is charged when that spike occurs and discharges linearly thereafter. A postsynaptic spike samples the charge remaining on the capacitor, passes it through an exponential function, and dumps the resultant charge into the integrator. This charge decays linearly thereafter. At the time of the postsynaptic spike, the SRAM, a cross-coupled inverter pair, reads the voltage on the integrator's capacitor. If it exceeds a threshold, the SRAM switches state from depressed to potentiated ($\sim$LTD goes high and $\sim$LTP goes low). The depression side of the STDP circuit is exactly symmetric, except that it responds to postsynaptic activation followed by presynaptic activation and switches the SRAM's state from potentiated to depressed ($\sim$LTP goes high and $\sim$LTD goes low). When the SRAM is in the potentiated state, the presynaptic

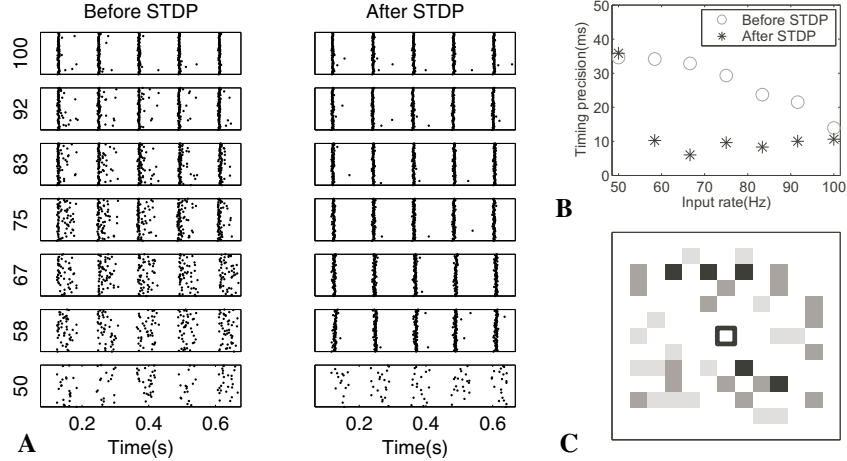

Figure 4: Plasticity enhanced phase-coding. **A** Spike rasters of 81 neurons (9 by 9 cluster) display synchrony over a two-fold range of input rates after STDP. **B** The degree of enhancement is quantified by timing precision. **C** Each neuron (center box) sends synapses to (dark gray) and receives synapses from (light gray) twenty-one randomly chosen neighbors up to five nodes away (black indicates both connections).

spike activates the principal neuron's synapse; otherwise the spike has no effect.

We characterized the STDP circuit by activating a plastic synapse and a fixed synapse–which elicits a spike at different relative times. We repeated this pairing at 16Hz. We counted the number of pairings required to potentiate (or depress) the synapse. Based on this count, we calculated the efficacy of each pairing as the inverse number of pairings required (Figure 3B). For example, if twenty pairings were required to potentiate the synapse, the efficacy of that pre-before-post time-interval was one twentieth. The efficacy of both potentiation and depression are fit by exponentials with time constants of 11.4ms and 94.9ms, respectively. This behavior is similar to that observed in the hippocampus: potentiation has a shorter time constant and higher maximum efficacy than depression [3].

## 4 Recurrent Network

We carried out an experiment designed to test the STDP circuit's ability to compensate for variability in spike timing through PEP. Each neuron received recurrent connections from 21 randomly selected neurons within an 11 by 11 neighborhood centered on itself (see Figure 4C). Conversely, it made recurrent connections to randomly chosen neurons within the same neighborhood. These connections were mediated by STDP circuits, initialized to the depressed state. We chose a 9 by 9 cluster of neurons and delivered spikes at a mean rate of 50 to 100Hz to each one (dropping spikes with a probability of 0.75 to 0.5 from a regular 200Hz train) and provided common theta inhibition as before.

We compared the variability in spike timing after five seconds of learning with the initial distribution. Phase coding was enhanced after STDP (Figure 4A). Before STDP, spike timing among neurons was highly variable (except for the very highest input rate). After STDP, variability was virtually eliminated (except for the very lowest input rate). Initially, the variability, characterized by timing precision, was inversely related to the input rate, decreasing from 34 to 13ms. After five seconds of STDP, variability decreased and was largely independent of input rate, remaining below 11ms.

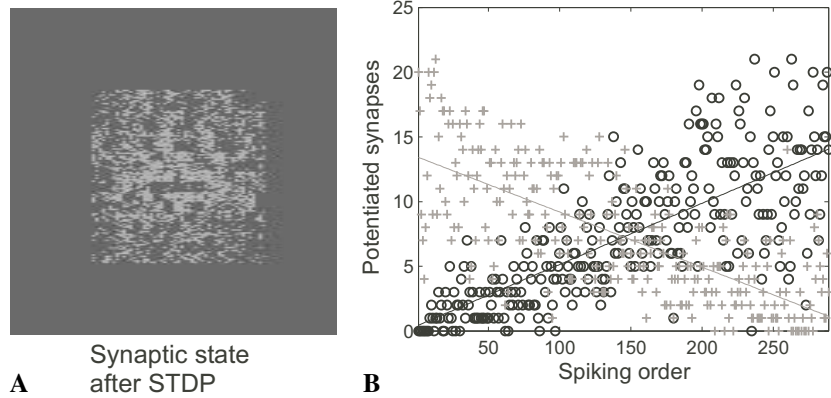

A       Synaptic state after STDP       B

Figure 5: Compensating for variability. **A** Some synapses (dots) become potentiated (light) while others remain depressed (dark) after STDP. **B** The number of potentiated synapses neurons make (pluses) and receive (circles) is negatively (r = -0.71) and positively (r = 0.76) correlated to their rank in the spiking order, respectively.

Comparing the number of potentiated synapses each neuron made or received with its excitability confirmed the PEP hypothesis (i.e., leading neurons provide additional synaptic current to lagging neurons via potentiated recurrent synapses). In this experiment, to eliminate variability due to noise (as opposed to excitability), we provided a 17 by 17 cluster of neurons with a regular 200Hz excitatory input. Theta inhibition was present as before and all synapses were initialized to the depressed state. After 10 seconds of STDP, a large fraction of the synapses were potentiated (Figure 5A). When the number of potentiated synapses each neuron made or received was plotted versus its rank in spiking order (Figure 5B), a clear correlation emerged (r = -0.71 or 0.76, respectively). As expected, neurons that spiked early made more and received fewer potentiated synapses. In contrast, neurons that spiked late made fewer and received more potentiated synapses.

## 5 Pattern Completion

After STDP, we found that the network could recall an entire pattern given a subset, thus the same mechanisms that compensated for variability and noise could also compensate for lack of information. We chose a 9 by 9 cluster of neurons as our pattern and delivered a poisson-like spike train with mean rate of 67Hz to each one as in the first experiment. Theta inhibition was present as before and all synapses were initialized to the depressed state. Before STDP, we stimulated a subset of the pattern and only neurons in that subset spiked (Figure 6A). After five seconds of STDP, we stimulated the same subset again. This time they recruited spikes from other neurons in the pattern, completing it (Figure 6B).

Upon varying the fraction of the pattern presented, we found that the fraction recalled increased faster than the fraction presented. We selected subsets of the original pattern randomly, varying the fraction of neurons chosen from 0.1 to 1.0 (ten trials for each). We classified neurons as active if they spiked in the two second period over which we recorded. Thus, we characterized PEP's pattern-recall performance as a function of the probability that the pattern in question's neurons are activated (Figure 6C). At a fraction of 0.50 presented, nearly all of the neurons in the pattern are consistently activated (0.91±0.06), showing robust pattern completion. We fitted the recall performance with a sigmoid that reached 0.50 recall fraction with an input fraction of 0.30. No spurious neurons were activated during any trials.

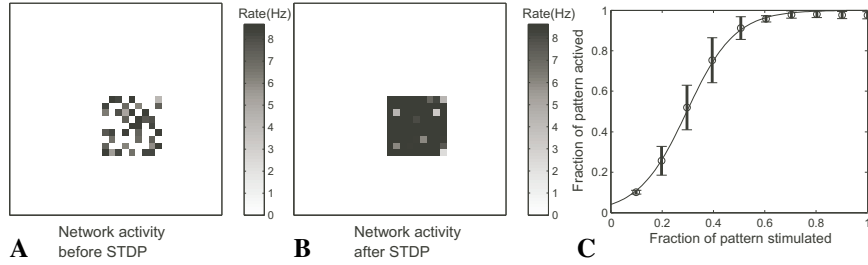

Figure 6: Associative recall. **A** Before STDP, half of the neurons in a pattern are stimulated; only they are activated. **B** After STDP, half of the neurons in a pattern are stimulated, and all are activated. **C** The fraction of the pattern activated grows faster than the fraction stimulated.

## 6 Discussion

Our results demonstrate that PEP successfully compensates for graded variations in our silicon recurrent network using binary (on–off) synapses (in contrast with [8], where weights are graded). While our chip results are encouraging, variability was not eliminated in every case. In the case of the lowest input (50Hz), we see virtually no change (Figure 4A). We suspect the timing remains imprecise because, with such low input, neurons do not spike every theta cycle and, consequently, provide fewer opportunities for the STDP synapses to potentiate. This shortfall illustrates the system's limits; it can only compensate for variability within certain bounds, and only for activity appropriate to the PEP model.

As expected, STDP is the mechanism responsible for PEP. STDP potentiated recurrent synapses from leading neurons to lagging neurons, reducing the disparity among the diverse population of neurons. Even though the STDP circuits are themselves variable, with different efficacies and time constants, when using timing the sign of the weight-change is always correct (data not shown). For this reason, we chose STDP over other more physiological implementations of plasticity, such as membrane-voltage-dependent plasticity (MVDP), which has the capability to learn with graded voltage signals [9], such as those found in active dendrites, providing more computational power [10].

Previously, we investigated a MVDP circuit, which modeled a voltage-dependent NMDA-receptor-gated synapse [11]. It potentiated when the calcium current analog exceeded a threshold, which was designed to occur only during a dendritic action potential. This circuit produced behavior similar to STDP, implying it could be used in PEP. However, it was sensitive to variability in the NMDA and potentiation thresholds, causing a fraction of the population to potentiate anytime the synapse received an input and another fraction to never potentiate, rendering both subpopulations useless. Therefore, the simpler, less biophysical STDP circuit won out over the MVDP circuit: In our system *timing is everything*.

Associative storage and recall naturally emerge in the PEP network when synapses between neurons coactivated by a pattern are potentiated. These synapses allow neurons to recruit their peers when a subset of the pattern is presented, thereby completing the pattern. However, this form of pattern storage and completion differs from Hopfield's attractor model [12] . Rather than forming symmetric, recurrent neuronal circuits, our recurrent network forms asymmetric circuits in which neurons make connections exclusively to less excitable neurons in the pattern. In both the poisson-like and regular cases (Figures 4 & 5), only about six percent of potentiated connections were reciprocated, as expected by chance. We plan to investigate the storage capacity of this asymmetric form of associative memory.

Our system lends itself to modeling brain regions that use precise spike timing, such as

the hippocampus. We plan to extend the work presented to store and recall sequences of patterns, as the hippocampus is hypothesized to do. Place cells that represent different locations spike at different phases of the theta cycle, in relation to the distance to their preferred locations. This sequential spiking will allow us to link patterns representing different locations in the order those locations are visited, thereby realizing episodic memory.

We propose PEP as a candidate neural mechanism for information coding and storage in the hippocampal system. Observations from the CA1 region of the hippocampus suggest that basal dendrites (which primarily receive excitation from recurrent connections) support submillisecond timing precision, consistent with PEP [13]. We have shown, in a silicon model, PEP's ability to exploit such fast recurrent connections to sharpen timing precision as well as to associatively store and recall patterns.

## Acknowledgments

We thank Joe Lin for assistance with chip generation. The Office of Naval Research funded this work (Award No. N000140210468).

## References

[1] O'Keefe J. & Recce M.L. (1993). Phase relationship between hippocampal place units and the EEG theta rhythm. *Hippocampus* **3**(3):317-330.

[2] Mehta M.R., Lee A.K. & Wilson M.A. (2002) Role of experience and oscillations in transforming a rate code into a temporal code. *Nature* **417**(6890):741-746.

[3] Bi G.Q. & Wang H.X. (2002) Temporal asymmetry in spike timing-dependent synaptic plasticity. *Physiology & Behavior* **77**:551-555.

[4] Rodriguez-Vazquez, A., Linan, G., Espejo S. & Dominguez-Castro R. (2003) Mismatch-induced trade-offs and scalability of analog preprocessing visual microprocessor chips. *Analog Integrated Circuits and Signal Processing* **37**:73-83.

[5] Boahen K.A. (2000) Point-to-point connectivity between neuromorphic chips using address events. *IEEE Transactions on Circuits and Systems II* **47**:416-434.

[6] Culurciello E.R., Etienne-Cummings R. & Boahen K.A. (2003) A biomorphic digital image sensor. *IEEE Journal of Solid State Circuits* **38**:281-294.

[7] Bofill A., Murray A.F & Thompson D.P. (2005) Citcuits for VLSI Implementation of Temporally Asymmetric Hebbian Learning. In: *Advances in Neural Information Processing Systems 14*, MIT Press, 2002.

[8] Cameron K., Boonsobhak V., Murray A. & Renshaw D. (2005) Spike timing dependent plasticity (STDP) can ameliorate process variations in neuromorphic VLSI. *IEEE Transactions on Neural Networks* **16**(6):1626-1627.

[9] Chicca E., Badoni D., Dante V., D'Andreagiovanni M., Salina G., Carota L., Fusi S. & Del Giudice P. (2003) A VLSI recurrent network of integrate-and-fire neurons connected by plastic synapses with long-term memory. *IEEE Transaction on Neural Networks* **14**(5):1297-1307.

[10] Poirazi P., & Mel B.W. (2001) Impact of active dendrites and structural plasticity on the memory capacity of neural tissue. *Neuron* **29**(3)779-796.

[11] Arthur J.V. & Boahen K. (2004) Recurrently connected silicon neurons with active dendrites for one-shot learning. In: *IEEE International Joint Conference on Neural Networks* **3**, pp.1699-1704.

[12] Hopfield J.J. (1984) Neurons with graded response have collective computational properties like those of two-state neurons. *Proceedings of the National Academy of Science* **81**(10):3088-3092.

[13] Ariav G., Polsky A. & Schiller J. (2003) Submillisecond precision of the input-output transformation function mediated by fast sodium dendritic spikes in basal dendrites of CA1 pyramidal neurons. *Journal of Neuroscience* **23**(21):7750-7758.